# Robust Multi-Class Gaussian Process Classification

**Daniel Hernández-Lobato**
ICTEAM - Machine Learning Group
Université catholique de Louvain
Place Sainte Barbe, 2
Louvain-La-Neuve, 1348, Belgium
danielhernandezlobato@gmail.com

**José Miguel Hernández-Lobato**
Department of Engineering
University of Cambridge
Trumpington Street, Cambridge
CB2 1PZ, United Kingdom
jmh233@eng.cam.ac.uk

**Pierre Dupont**
ICTEAM - Machine Learning Group
Université catholique de Louvain
Place Sainte Barbe, 2
Louvain-La-Neuve, 1348, Belgium
pierre.dupont@uclouvain.be

## Abstract

Multi-class Gaussian Process Classifiers (MGPCs) are often affected by over-fitting problems when labeling errors occur far from the decision boundaries. To prevent this, we investigate a robust MGPC (RMGPC) which considers labeling errors independently of their distance to the decision boundaries. Expectation propagation is used for approximate inference. Experiments with several datasets in which noise is injected in the labels illustrate the benefits of RMGPC. This method performs better than other Gaussian process alternatives based on considering latent Gaussian noise or heavy-tailed processes. When no noise is injected in the labels, RMGPC still performs equal or better than the other methods. Finally, we show how RMGPC can be used for successfully identifying data instances which are difficult to classify correctly in practice.

## 1 Introduction

Multi-class Gaussian process classifiers (MGPCs) are a Bayesian approach to non-parametric multi-class classification with the advantage of producing probabilistic outputs that measure uncertainty in the predictions [1]. MGPCs assume that there are some latent functions (one per class) whose value at a certain location is related by some rule to the probability of observing a specific class there. The prior for each of these latent functions is specified to be a Gaussian process. The task of interest is to make inference about the latent functions using Bayes' theorem. Nevertheless, exact Bayesian inference in MGPCs is typically intractable and one has to rely on approximate methods. Approximate inference can be implemented using Markov-chain Monte Carlo sampling, the Laplace approximation or expectation propagation [2, 3, 4, 5].

A problem of MGPCs is that, typically, the assumed rule that relates the values of the latent functions with the different classes does not consider the possibility of observing errors in the labels of the data, or at most, only considers the possibility of observing errors near the decision boundaries of the resulting classifier [1]. The consequence is that over-fitting can become a serious problem when errors far from these boundaries are observed in practice. A notable exception is found in the binary classification case when the labeling rule suggested in [6] is used. Such rule considers the possibility of observing errors independently of their distance to the decision boundary [7, 8]. However, the generalization of this rule to the multi-class case is difficult. Existing generalizations

are in practice simplified so that the probability of observing errors in the labels is zero [3]. Labeling errors in the context of MGPCs are often accounted for by considering that the latent functions of the MGPC are contaminated with additive Gaussian noise [1]. Nevertheless, this approach has again the disadvantage of considering only errors near the decision boundaries of the resulting classifier and is expected to lead to over-fitting problems when errors are actually observed far from the boundaries. Finally, some authors have replaced the underlying Gaussian processes of the MGPC with heavy-tailed processes [9]. These processes have marginal distributions with heavier tails than those of a Gaussian distribution and are in consequence expected to be more robust to labeling errors far from the decision boundaries.

In this paper we investigate a robust MGPC (RMGPC) that addresses labeling errors by introducing a set of binary latent variables. One latent variable for each data instance. These latent variables indicate whether the assumed labeling rule is satisfied for the associated instances or not. If such rule is not satisfied for a given instance, we consider that the corresponding label has been randomly selected with uniform probability among the possible classes. This is used as a back-up mechanism to explain data instances that are highly unlikely to stem from the assumed labeling rule. The resulting likelihood function depends only on the total number of errors, and not on the distances of these errors to the decision boundaries. Thus, RMGPC is expected to be fairly robust when the data contain noise in the labels. In this model, expectation propagation (EP) can be used to efficiently carry out approximate inference [10]. The cost of EP is $\mathcal{O}(ln^3)$, where $n$ is the number of training instances and $l$ is the number of different classes. RMGPC is evaluated in four datasets extracted from the UCI repository [11] and from other sources [12]. These experiments show the beneficial properties of the proposed model in terms of prediction performance. When labeling noise is introduced in the data, RMGPC outperforms other MGPC approaches based on considering latent Gaussian noise or heavy-tailed processes. When there is no noise in the data, RMGPC performs better or equivalent to these alternatives. Extra experiments also illustrate the utility of RMGPC to identify data instances that are unlikely to stem from the assumed labeling rule.

The organization of the rest of the manuscript is as follows: Section 2 introduces the RMGPC model. Section 3 describes how expectation propagation can be used for approximate Bayesian inference. Then, Section 4 evaluates and compares the predictive performance of RMGPC. Finally, Section 5 summarizes the conclusions of the investigation.

## 2   Robust Multi-Class Gaussian Process Classification

Consider $n$ training instances in the form of a collection of feature vectors $\mathbf{X} = \{\mathbf{x}_1, \ldots, \mathbf{x}_n\}$ with associated labels $\mathbf{y} = \{y_1, \ldots, y_n\}$, where $y_i \in \mathcal{C} = \{1, \ldots, l\}$ and $l$ is the number of classes. We follow [3] and assume that, in the noise free scenario, the predictive rule for $y_i$ given $\mathbf{x}_i$ is

$$y_i = \arg\max_k f_k(\mathbf{x}_i)\,, \tag{1}$$

where $f_1, \ldots, f_l$ are unknown latent functions that have to be estimated. The prediction rule given by (1) is unlikely to hold always in practice. For this reason, we introduce a set of binary latent variables $\mathbf{z} = \{z_1, \ldots, z_n\}$, one per data instance, to indicate whether (1) is satisfied ($z_i = 0$) or not ($z_i = 1$). In this latter case, the pair $(\mathbf{x}_i, y_i)$ is considered to be an outlier and, instead of assuming that $y_i$ is generated by (1), we assume that $\mathbf{x}_i$ is assigned a random class sampled uniformly from $\mathcal{C}$. This is equivalent to assuming that $f_1, \ldots, f_l$ have been contaminated with an infinite amount of noise and serves as a *back-up* mechanism to explain observations which are highly unlikely to originate from (1). The likelihood function for $\mathbf{f} = (f_1(\mathbf{x}_1), f_1(\mathbf{x}_2) \ldots, f_1(\mathbf{x}_n), f_2(\mathbf{x}_1), f_2(\mathbf{x}_2) \ldots, f_2(\mathbf{x}_n), \ldots, f_l(\mathbf{x}_1), f_l(\mathbf{x}_2), \ldots, f_l(\mathbf{x}_n))^{\mathrm{T}}$ given $\mathbf{y}$, $\mathbf{X}$ and $\mathbf{z}$ is

$$\mathcal{P}(\mathbf{y}|\mathbf{X}, \mathbf{z}, \mathbf{f}) = \prod_{i=1}^n \left[ \prod_{k \neq y_i} \Theta(f_{y_i}(\mathbf{x}_i) - f_k(\mathbf{x}_i)) \right]^{1-z_i} \left[ \frac{1}{l} \right]^{z_i}, \tag{2}$$

where $\Theta(\cdot)$ is the Heaviside step function. In (2), the contribution to the likelihood of each instance $(\mathbf{x}_i, y_i)$ is a a mixture of two terms: A first term equal to $\prod_{k \neq y_i} \Theta(f_{y_i}(\mathbf{x}_i) - f_k(\mathbf{x}_i))$ and a second term equal to $1/l$. The mixing coefficient is the prior probability of $z_i = 1$. Note that only the first term actually depends on the accuracy of $\mathbf{f}$. In particular, it takes value 1 when the corresponding instance is correctly classified using (1) and 0 otherwise. Thus, the likelihood function described in

(2) considers only the total number of prediction errors made by $\mathbf{f}$ and not the distance of these errors to the decision boundary. The consequence is that (2) is expected to be robust when the observed data contain labeling errors far from the decision boundaries.

We do not have any preference for a particular instance to be considered an outlier. Thus, $\mathbf{z}$ is set to follow *a priori* a factorizing multivariate Bernoulli distribution:

$$\mathcal{P}(\mathbf{z}|\rho) = \text{Bern}(\mathbf{z}|\rho) = \prod_{i=1}^{n} \rho^{z_i}(1-\rho)^{1-z_i} \,, \tag{3}$$

where $\rho$ is the prior fraction of training instances expected to be outliers. The prior for $\rho$ is set to be a conjugate beta distribution, namely

$$\mathcal{P}(\rho) = \text{Beta}(\rho|a_0, b_0) = \frac{\rho^{a_0-1}(1-\rho)^{b_0-1}}{\text{B}(a_0, b_0)} \,, \tag{4}$$

where $\text{B}(\cdot, \cdot)$ is the beta function and $a_0$ and $b_0$ are free hyper-parameters. The values of $a_0$ and $b_0$ do not have a big impact on the final model provided that they are consistent with the prior belief that most of the observed data are labeled using (1) ($b_0 > a_0$) and that they are small such that (4) is not too constraining. We suggest $a_0 = 1$ and $b_0 = 9$.

As in [3], the prior for $f_1, \ldots, f_l$ is set to be a product of Gaussian processes with means equal to $\mathbf{0}$ and covariance matrices $\mathbf{K}_1, \ldots, \mathbf{K}_l$, as computed by $l$ covariance functions $c_1(\cdot, \cdot), \ldots, c_l(\cdot, \cdot)$:

$$\mathcal{P}(\mathbf{f}) = \prod_{k=1}^{l} \mathcal{N}(\mathbf{f}_k|\mathbf{0}, \mathbf{K}_k) \tag{5}$$

where $\mathcal{N}(\cdot|\boldsymbol{\mu}, \boldsymbol{\Sigma})$ denotes a multivariate Gaussian density with mean vector $\boldsymbol{\mu}$ and covariance matrix $\boldsymbol{\Sigma}$, $\mathbf{f}$ is defined as in (2) and $\mathbf{f}_k = (f_k(\mathbf{x}_1), f_k(\mathbf{x}_2), \ldots, f_k(\mathbf{x}_n))^{\text{T}}$, for $k = 1, \ldots, l$.

## 2.1 Inference, Prediction and Outlier Identification

Given the observed data $\mathbf{X}$ and $\mathbf{y}$, we make inference about $\mathbf{f}$, $\mathbf{z}$ and $\rho$ using Bayes' theorem:

$$\mathcal{P}(\rho, \mathbf{z}, \mathbf{f}|\mathbf{y}, \mathbf{X}) = \frac{\mathcal{P}(\mathbf{y}|\mathbf{X}, \mathbf{z}, \mathbf{f})\mathcal{P}(\mathbf{z}|\rho)\mathcal{P}(\rho)\mathcal{P}(\mathbf{f})}{\mathcal{P}(\mathbf{y}|\mathbf{X})} \,, \tag{6}$$

where $\mathcal{P}(\mathbf{y}|\mathbf{X})$ is the model evidence, a constant useful to perform model comparison under a Bayesian setting [13]. The posterior distribution and the likelihood function can be used to compute a predictive distribution for the label $y_\star \in \mathcal{C}$ associated to a new observation $\mathbf{x}_\star$:

$$\mathcal{P}(y_\star|\mathbf{x}_\star, \mathbf{y}, \mathbf{X}) = \sum_{\mathbf{z}, z_\star} \int \mathcal{P}(y_\star|\mathbf{x}_\star, z_\star, \mathbf{f}_\star)\mathcal{P}(z_\star|\rho)\mathcal{P}(\mathbf{f}_\star|\mathbf{f})\mathcal{P}(\rho, \mathbf{z}, \mathbf{f}|\mathbf{y}, \mathbf{X}) \, d\mathbf{f} \, d\mathbf{f}_\star \, d\rho , \tag{7}$$

where $\mathbf{f}_\star = (f_1(\mathbf{x}_\star), \ldots, f_l(\mathbf{x}_\star))^{\text{T}}$, $\mathcal{P}(y_\star|\mathbf{x}_\star, z_\star, \mathbf{f}_\star) = \prod_{k \neq y_\star} \Theta(f_k(\mathbf{x}_\star) - f_{y_\star}(\mathbf{x}_\star))^{1-z_\star}(1/l)^{z_\star}$, $\mathcal{P}(z_\star|\rho) = \rho^{z_\star}(1-\rho)^{1-z_\star}$ and $\mathcal{P}(\mathbf{f}_\star|\mathbf{f})$ is a product of $l$ conditional Gaussians with zero mean and covariance matrices given by the covariance functions of $\mathbf{K}_1, \ldots, \mathbf{K}_l$. The posterior for $\mathbf{z}$ is

$$\mathcal{P}(\mathbf{z}|\mathbf{y}, \mathbf{X}) = \int \mathcal{P}(\rho, \mathbf{z}, \mathbf{f}|\mathbf{y}, \mathbf{X}) d\mathbf{f} d\rho \,. \tag{8}$$

This distribution is useful to compute the posterior probability that the $i$-th training instance is an outlier, *i.e.*, $\mathcal{P}(z_i = 1|\mathbf{y}, \mathbf{X})$. For this, we only have to marginalize (8) with respect to all the components of $\mathbf{z}$ except $z_i$. Unfortunately, the exact computation of (6), (7) and $\mathcal{P}(z_i = 1|\mathbf{y}, \mathbf{X})$ is intractable for typical classification problems. Nevertheless, these expressions can be approximated using expectation propagation [10].

## 3 Expectation Propagation

The joint probability of $\mathbf{f}$, $\mathbf{z}$, $\rho$ and $\mathbf{y}$ given $\mathbf{X}$ can be written as the product of $l(n+1)+1$ factors:

$$\mathcal{P}(\mathbf{f}, \mathbf{z}, \rho, \mathbf{y}|\mathbf{X}) = \mathcal{P}(\mathbf{y}|\mathbf{X}, \mathbf{z}, \mathbf{f})\mathcal{P}(\mathbf{z}|\rho)\mathcal{P}(\rho)\mathcal{P}(\mathbf{f})$$

$$= \left[\prod_{i=1}^{n} \prod_{k \neq y_i} \psi_{ik}(\mathbf{f}, \mathbf{z}, \rho)\right] \left[\prod_{i=1}^{n} \psi_i(\mathbf{f}, \mathbf{z}, \rho)\right] \psi_\rho(\mathbf{f}, \mathbf{z}, \rho) \left[\prod_{k=1}^{l} \psi_k(\mathbf{f}, \mathbf{z}, \rho)\right] , \tag{9}$$

where each factor has the following form:

$$\psi_{ik}(\mathbf{f}, \mathbf{z}, \rho) = \Theta(f_{y_i}(\mathbf{x}_i) - f_k(\mathbf{x}_i))^{1-z_i}(l^{-\frac{1}{l-1}})^{z_i}, \qquad \psi_i(\mathbf{f}, \mathbf{z}, \rho) = \rho^{z_i}(1-\rho)^{1-z_i},$$

$$\psi_\rho(\mathbf{f}, \mathbf{z}, \rho) = \frac{\rho^{a_0-1}(1-\rho)^{b_0-1}}{\mathrm{B}(a_0, b_0)}, \qquad \psi_k(\mathbf{f}, \mathbf{z}, \rho) = \mathcal{N}(\mathbf{f}_k | \mathbf{0}, \mathbf{K}_k). \qquad (10)$$

Let $\Psi$ be the set that contains all these exact factors. Expectation propagation (EP) approximates each $\psi \in \Psi$ using a corresponding simpler factor $\tilde\psi$ such that

$$\left[\prod_{i=1}^{n}\prod_{k\neq y_i}\psi_{ik}\right]\left[\prod_{i=1}^{n}\psi_i\right]\psi_\rho\left[\prod_{k=1}^{l}\psi_k\right] \approx \left[\prod_{i=1}^{n}\prod_{k\neq y_i}\tilde\psi_{ik}\right]\left[\prod_{i=1}^{n}\tilde\psi_i\right]\tilde\psi_\rho\left[\prod_{k=1}^{l}\tilde\psi_k\right]. \qquad (11)$$

In (11) the dependence of the exact and the approximate factors on $\mathbf{f}$, $\mathbf{z}$ and $\rho$ has been removed to improve readability. The approximate factors $\tilde\psi$ are constrained to belong to the same family of exponential distributions, but they do not have to integrate to one. Once normalized with respect to $\mathbf{f}$, $\mathbf{z}$ and $\rho$, (9) becomes the exact posterior distribution (6). Similarly, the normalized product of the approximate factors becomes an approximation to the posterior distribution:

$$\mathcal{Q}(\mathbf{f}, \mathbf{z}, \rho) = \frac{1}{Z}\left[\prod_{i=1}^{n}\prod_{k\neq y_i}\tilde\psi_{ik}(\mathbf{f}, \mathbf{z}, \rho)\right]\left[\prod_{i=1}^{n}\tilde\psi_i(\mathbf{f}, \mathbf{z}, \rho)\right]\tilde\psi_\rho(\mathbf{f}, \mathbf{z}, \rho)\left[\prod_{k=1}^{l}\tilde\psi_k(\mathbf{f}, \mathbf{z}, \rho)\right], \qquad (12)$$

where $Z$ is a normalization constant that approximates $\mathcal{P}(\mathbf{y}|\mathbf{X})$. Exponential distributions are closed under product and division operations. Therefore, $\mathcal{Q}$ has the same form as the approximate factors and $Z$ can be readily computed. In practice, the form of $\mathcal{Q}$ is selected first, and the approximate factors are then constrained to have the same form as $\mathcal{Q}$. For each approximate factor $\tilde\psi$ define $\mathcal{Q}^{\backslash\tilde\psi} \propto \mathcal{Q}/\tilde\psi$ and consider the corresponding exact factor $\psi$. EP iteratively updates each $\tilde\psi$, one by one, so that the Kullback-Leibler (KL) divergence between $\psi\mathcal{Q}^{\backslash\tilde\psi}$ and $\tilde\psi\mathcal{Q}^{\backslash\tilde\psi}$ is minimized. The EP algorithm involves the following steps:

1. Initialize all the approximate factors $\tilde\psi$ and the posterior approximation $\mathcal{Q}$ to be uniform.

2. Repeat until $\mathcal{Q}$ converges:

    (a) Select an approximate factor $\tilde\psi$ to refine and compute $\mathcal{Q}^{\backslash\tilde\psi} \propto \mathcal{Q}/\tilde\psi$.

    (b) Update the approximate factor $\tilde\psi$ so that $\mathrm{KL}(\psi\mathcal{Q}^{\backslash\tilde\psi}||\tilde\psi\mathcal{Q}^{\backslash\tilde\psi})$ is minimized.

    (c) Update the posterior approximation $\mathcal{Q}$ to the normalized version of $\tilde\psi\mathcal{Q}^{\backslash\tilde\psi}$.

3. Evaluate $Z \approx \mathcal{P}(\mathbf{y}|\mathbf{X})$ as the integral of the product of all the approximate factors.

The optimization problem in step 2-(b) is convex with a single global optimum. The solution to this problem is found by matching sufficient statistics between $\psi\mathcal{Q}^{\backslash\tilde\psi}$ and $\tilde\psi\mathcal{Q}^{\backslash\tilde\psi}$. EP is not guaranteed to converge globally but extensive empirical evidence shows that most of the times it converges to a fixed point [10]. Non-convergence can be prevented by *damping* the EP updates [14]. Damping is a standard procedure and consists in setting $\tilde\psi = [\tilde\psi_{\mathrm{new}}]^\epsilon[\tilde\psi_{\mathrm{old}}]^{1-\epsilon}$ in step 2-(b), where $\tilde\psi_{\mathrm{new}}$ is the updated factor and $\tilde\psi_{\mathrm{old}}$ is the factor before the update. $\epsilon \in [0, 1]$ is a parameter which controls the amount of damping. When $\epsilon = 1$, the standard EP update operation is recovered. When $\epsilon = 0$, no update of the approximate factors occurs. In our experiments $\epsilon = 0.5$ gives good results and EP seems to always converge to a stationary solution. EP has shown good overall performance when compared to other methods in the task of classification with binary Gaussian processes [15, 16].

## 3.1 The Posterior Approximation

The posterior distribution (6) is approximated by a distribution $\mathcal{Q}$ in the exponential family:

$$\mathcal{Q}(\mathbf{f}, \mathbf{z}, \rho) = \mathrm{Bern}(\mathbf{z}|\mathbf{p})\mathrm{Beta}(\rho|a, b)\prod_{k=1}^{l}\mathcal{N}(\mathbf{f}_k|\boldsymbol{\mu}_k, \boldsymbol{\Sigma}_k), \qquad (13)$$

where $\mathcal{N}(\cdot|, \boldsymbol{\mu}, \boldsymbol{\Sigma})$ is a multivariate Gaussian distribution with mean $\boldsymbol{\mu}$ and covariance matrix $\boldsymbol{\Sigma}$; $\mathrm{Beta}(\cdot|a, b)$ is a beta distribution with parameters $a$ and $b$; and $\mathrm{Bern}(\cdot|\mathbf{p})$ is a multivariate Bernoulli

distribution with parameter vector $\mathbf{p}$. The parameters $\boldsymbol{\mu}_k$ and $\boldsymbol{\Sigma}_k$ for $k = 1, \ldots, l$ and $\mathbf{p}$, $a$ and $b$ are estimated by EP. Note that $\mathcal{Q}$ factorizes with respect to $\mathbf{f}_k$ for $k = 1, \ldots, l$. This makes the cost of the EP algorithm linear in $l$, the total number of classes. More accurate approximations can be obtained at a cubic cost in $l$ by considering correlations among the $\mathbf{f}_k$. The choice of (13) also makes all the required computations tractable and provides good results in Section 4.

The approximate factors must have the same functional form as $\mathcal{Q}$ but they need not be normalized. However, the exact factors $\psi_{ik}$ with $i = 1, \ldots, n$ and $k \neq y_i$, corresponding to the likelihood, (2), only depend on $f_k(\mathbf{x}_i)$, $f_{y_i}(\mathbf{x}_i)$ and $z_i$. Thus, the beta part of the corresponding approximate factors can be removed and the multivariate Gaussian distributions simplify to univariate Gaussians. Specifically, the approximate factors $\tilde{\psi}_{ik}$ with $i = 1, \ldots, n$ and $k \neq y_i$ are:

$$\tilde{\psi}_{ik}(\mathbf{f}, \mathbf{z}, \rho) = \tilde{s}_{ik} \exp \left\{ -\frac{1}{2} \left[ \frac{(f_k(\mathbf{x}_i) - \tilde{\mu}_{ik})^2}{\tilde{\nu}_{ik}} + \frac{(f_{y_i}(\mathbf{x}_i) - \tilde{\mu}_{ik}^{y_i})^2}{\tilde{\nu}_{ik}^{y_i}} \right] \right\} \tilde{p}_{ik}^{z_i} (1 - \tilde{p}_{ik})^{1 - z_i} , \quad (14)$$

where $\tilde{s}_{ik}$, $\tilde{p}_{ik}$, $\tilde{\mu}_{ik}$, $\tilde{\nu}_{ik}$, $\tilde{\mu}_{ik}^{y_i}$ and $\tilde{\nu}_{ik}^{y_i}$ are free parameters to be estimated by EP. Similarly, the exact factors $\psi_i$, with $i = 1, \ldots, n$, corresponding to the prior for the latent variables $\mathbf{z}$, (3), only depend on $\rho$ and $z_i$. Thus, the Gaussian part of the corresponding approximate factors can be removed and the multivariate Bernoulli distribution simplifies to a univariate Bernoulli. The resulting factors are:

$$\tilde{\psi}_i(\mathbf{f}, \mathbf{z}, \rho) = \tilde{s}_i \rho^{\tilde{a}_i - 1} (1 - \rho)^{\tilde{b}_i - 1} \tilde{p}_i^{z_i} (1 - \tilde{p}_i)^{1 - z_i} , \quad (15)$$

for $i = 1, \ldots, n$, where $\tilde{s}_i$, $\tilde{a}_i$, $\tilde{b}_i$, $\tilde{p}_i$ are free parameters to be estimated by EP. The exact factor $\psi_\rho$ corresponding to the prior for $\rho$, (4), need not be approximated, i.e., $\tilde{\psi}_\rho = \psi_\rho$. The same applies to the exact factors $\psi_k$, for $k = 1, \ldots, l$, corresponding to the priors for $\mathbf{f}_1, \ldots, \mathbf{f}_l$, (5). We set $\tilde{\psi}_k = \psi_k$ for $k = 1, \ldots, l$. All these factors $\tilde{\psi}_\rho$ and $\tilde{\psi}_k$, for $k = 1, \ldots, l$, need not be refined by EP.

### 3.2 The EP Update Operations

The approximate factors $\tilde{\psi}_{ik}$, for $i = 1, \ldots, n$ and $k \neq y_i$, corresponding to the likelihood, are refined in parallel, as in [17]. This notably simplifies the EP updates. In particular, for each $\tilde{\psi}_{ik}$ we compute $\mathcal{Q}^{\setminus \tilde{\psi}_{ik}}$ as in step 2-(a) of EP. Given each $\mathcal{Q}^{\setminus \tilde{\psi}_{ik}}$ and the exact factor $\psi_{ik}$, we update each $\tilde{\psi}_{ik}$. Then, $\mathcal{Q}^{\text{new}}$ is re-computed as the normalized product of all the approximate factors. Preliminary experiments indicate that parallel and sequential updates converge to the same solution. The remaining factors, i.e., $\tilde{\psi}_i$, for $i = 1, \ldots, n$, are updated sequentially, as in standard EP. Further details about all these EP updates are found in the supplementary material[1]. The cost of EP, assuming constant iterations until convergence, is $\mathcal{O}(ln^3)$. This is the cost of inverting $l$ matrices of size $n \times n$.

### 3.3 Model Evidence, Prediction and Outlier Identification

Once EP has converged, we can evaluate the approximation to the model evidence as the integral of the product of all the approximate terms. This gives the following result:

$$\log Z = B + \left[ \sum_{i=1}^{n} \log D_i \right] + \frac{1}{2} \left[ \sum_{k=1}^{l} C_k - \log |\mathbf{M}_k| \right] + \left[ \sum_{i=1}^{n} \left[ \sum_{k \neq y_i} \log \tilde{s}_{ik} \right] + \log \tilde{s}_i \right] , \quad (16)$$

where

$$D_i = \tilde{p}_i \left[ \prod_{k \neq y_i} \tilde{p}_{ik} \right] + (1 - \tilde{p}_i) \left[ \prod_{k \neq y_i} (1 - \tilde{p}_{ik}) \right] , \quad C_k = \boldsymbol{\mu}_k^{\mathrm{T}} \boldsymbol{\Sigma}_k^{-1} \boldsymbol{\mu}_k - \sum_{i=1}^{n} \tau_i^k ,$$

$$\tau_i^k = \begin{cases} \sum_{k \neq y_i} (\tilde{\mu}_{ik}^{y_i})^2 / \tilde{\nu}_{ik}^{y_i} & \text{if } k = y_i , \\ \tilde{\mu}_{ik}^2 / \tilde{\nu}_{ik} & \text{otherwise} , \end{cases} \qquad B = \log \mathrm{B}(a, b) - \log \mathrm{B}(a_0, b_0) , \quad (17)$$

and $\mathbf{M}_k = \boldsymbol{\Lambda}^k \mathbf{K}_k + \mathbf{I}$, with $\boldsymbol{\Lambda}^k$ a diagonal matrix defined as $\Lambda_{ii}^k = \sum_{k \neq y_i} (\tilde{\nu}_{ik}^{y_i})^{-1}$, if $y_i = k$, and $\Lambda_{ii}^k = \tilde{\nu}_{ik}^{-1}$ otherwise. It is possible to compute the gradient of $\log Z$ with respect to $\theta_{kj}$, i.e., the $j$-th

hyper-parameter of the $k$-th covariance function used to compute $\mathbf{K}_k$. Such gradient is useful to find the covariance functions $c_k(\cdot, \cdot)$, with $k = 1, \ldots, l$, that maximize the model evidence. Specifically, one can show that, if EP has converged, the gradient of the free parameters of the approximate factors with respect to $\theta_{kj}$ is zero [18]. Thus, the gradient of $\log Z$ with respect to $\theta_{kj}$ is

$$\frac{\partial \log Z}{\partial \theta_{kj}} = -\frac{1}{2}\text{trace}\left(\mathbf{M}_k^{-1}\mathbf{\Lambda}^k\frac{\partial \mathbf{K}_k}{\partial \theta_{kj}}\right) + \frac{1}{2}(\boldsymbol{v}^k)^{\mathrm{T}}(\mathbf{M}_k^{-1})^{\mathrm{T}}\frac{\partial \mathbf{K}_k}{\partial \theta_{kj}}\mathbf{M}_k^{-1}\boldsymbol{v}^k \,, \qquad (18)$$

where $\boldsymbol{v}^k = (b_1^k, b_2^k, \ldots, b_n^k)^{\mathrm{T}}$ with $b_i^k = \sum_{k \neq y_i} \tilde{\mu}_{ik}^{y_i}/\tilde{\nu}_{ik}^{y_i}$, if $k = y_i$, and $b_i^k = \tilde{\mu}_{ik}/\tilde{\nu}_{ik}$ otherwise.

The predictive distribution (7) can be approximated when the exact posterior is replaced by $\mathcal{Q}$:

$$\mathcal{P}(y_\star|\mathbf{x}_\star, \mathbf{y}, \mathbf{X}) \approx \frac{\overline{\rho}}{l} + (1 - \overline{\rho}) \int \mathcal{N}(u|m_{y_\star}, v_{y_\star}) \prod_{k \neq y_\star} \Phi\left(\frac{u - m_k}{\sqrt{v_k}}\right) du \,, \qquad (19)$$

where $\Phi(\cdot)$ is the cumulative probability function of a standard Gaussian distribution and

$$\overline{\rho} = a/(a + b) \,, \quad m_k = (\mathbf{k}_k^\star)^{\mathrm{T}}\mathbf{K}_k^{-1}\mathbf{M}_k\boldsymbol{v}^k \,, \quad v_k = \kappa_k^\star - (\mathbf{k}_k^\star)^{\mathrm{T}}\left(\mathbf{K}_k^{-1} - \mathbf{K}_k^{-1}\mathbf{\Sigma}_k\mathbf{K}_k^{-1}\right)\mathbf{k}_k^\star \,, \quad (20)$$

for $k = 1, \ldots, l$, with $\mathbf{k}_k^\star$ equal to the covariances between $\mathbf{x}_\star$ and $\mathbf{X}$, and with $\kappa_k^\star$ equal to the corresponding variance at $\mathbf{x}_\star$, as computed by $c_k(\cdot, \cdot)$. There is no closed form expression for the integral in (19). However, it can be easily approximated by a one-dimensional quadrature.

The posterior (8) of $\mathbf{z}$ can be similarly approximated by marginalizing $\mathcal{Q}$ with respect to $\rho$ and $\mathbf{f}$:

$$\mathcal{P}(\mathbf{z}|\mathbf{y}, \mathbf{X}) \approx \text{Bern}(\mathbf{z}|\mathbf{p}) = \prod_{i=1}^{n}\left[p_i^{z_i}(1 - p_i)^{1-z_i}\right] \,, \qquad (21)$$

where $\mathbf{p} = (p_1, \ldots, p_n)^{\mathrm{T}}$. Each parameter $p_i$ of $\mathcal{Q}$, with $1 \leq i \leq n$, approximates $\mathcal{P}(z_i = 1|\mathbf{y}, \mathbf{X})$, *i.e.*, the posterior probability that the $i$-th training instance is an outlier. Thus, these parameters can be used to identify the data instances that are more likely to be outliers.

The cost of evaluating (16) and (18) is respectively $\mathcal{O}(ln^3)$ and $\mathcal{O}(n^3)$. The cost of evaluating (19) is $\mathcal{O}(ln^2)$ since $\mathbf{K}_k^{-1}$, with $k = 1, \ldots, l$, needs to be computed only once.

## 4 Experiments

The proposed Robust Multi-class Gaussian Process Classifier (RMGPC) is compared in several experiments with the Standard Multi-class Gaussian Process Classifier (SMGPC) suggested in [3]. SMGPC is a particular case of RMGPC which is obtained when $b_0 \to \infty$. This forces the prior distribution for $\rho$, (4), to be a delta centered at the origin, indicating that it is not possible to observe outliers. SMGPC explains data instances for which (1) is not satisfied in practice by considering Gaussian noise in the estimation of the functions $f_1, \ldots, f_l$, which is the typical approach found in the literature [1]. RMGPC is also compared in these experiments with the Heavy-Tailed Process Classifier (HTPC) described in [9]. In HTPC, the prior for each latent function $f_k$, with $k = 1, \ldots, l$, is a Gaussian Process that has been non-linearly transformed to have marginals that follow hyperbolic secant distributions with scale parameter $b_k$. The hyperbolic secant distribution has heavier tails than the Gaussian distribution and is expected to perform better in the presence of outliers.

### 4.1 Classification of Noisy Data

We carry out experiments on four datasets extracted from the UCI repository [11] and from other sources [12] to evaluate the predictive performance of RMGPC, SMGPC and HTPC when different fractions of outliers are present in the data[2]. These datasets are described in Table 1. All have multiple classes and a fairly small number $n$ of instances. We have selected problems with small $n$ because all the methods analyzed scale as $\mathcal{O}(n^3)$. The data for each problem are randomly split 100 times into training and test sets containing respectively $2/3$ and $1/3$ of the data. Furthermore, the labels of $\eta \in \{0\%, 5\%, 10\%, 20\%\}$ of the training instances are selected uniformly at random from $\mathcal{C}$. The data are normalized to have zero mean and unit standard deviation on the training set and

the average balanced class rate (BCR) of each method on the test set is reported for each value of $\eta$. The BCR of a method with prediction accuracy $a_k$ on those instances of class $k$ ($k = 1, \ldots, l$) is defined as $1/l \sum_{k=1}^{l} a_k$. BCR is preferred to prediction accuracy in datasets with unbalanced class distributions, which is the case for the datasets displayed in Table 1.

Table 1: Characteristics of the datasets used in the experiments.

| Dataset | # Instances | # Attributes | # Classes | # Source |
|---|---|---|---|---|
| New-thyroid | 215 | 5 | 3 | UCI |
| Wine | 178 | 13 | 3 | UCI |
| Glass | 214 | 9 | 6 | UCI |
| SVMguide2 | 319 | 20 | 3 | LIBSVM |

In our experiments, the different methods analyzed (RMGPC, SMGPC and HTPC) use the same covariance function for each latent function, *i.e.*, $c_k(\cdot, \cdot) = c(\cdot, \cdot)$, for $k = 1, \ldots, l$, where

$$c(\mathbf{x}_i, \mathbf{x}_j) = \exp \left\{ -\frac{1}{2\gamma} (\mathbf{x}_i - \mathbf{x}_j)^{\mathrm{T}} (\mathbf{x}_i - \mathbf{x}_j) \right\} \tag{22}$$

is a standard Gaussian covariance function with length-scale parameter $\gamma$. Preliminary experiments on the datasets analyzed show no significant benefit from considering a different covariance function for each latent function. The diagonal of the covariance matrices $\mathbf{K}_k$, for $k = 1, \ldots, l$, of SMGPC are also added an extra term equal to $\vartheta_k^2$ to account for latent Gaussian noise with variance $\vartheta_k^2$ around $f_k$ [1]. These extra terms are used by SMGPC to explain those instances that are unlikely to stem from (1). In both RMGPC and SMGPC the parameter $\gamma$ is found by maximizing (16) using a standard gradient ascent procedure. The same method is used for tuning the parameters $\vartheta_k$ in SMGPC. In HTPC an approximation to the model evidence is maximized with respect to $\gamma$ and the scale parameters $b_k$, with $k = 1, \ldots, l$, using also gradient ascent [9].

Table 2: Average BCR in % of each method for each problem, as a function of $\eta$.

| Dataset | RMGPC | SMGPC | HTPC | RMGPC | SMGPC | HTPC |
|---|---|---|---|---|---|---|
| | $\eta = 0\%$ | | | $\eta = 5\%$ | | |
| New-thyroid | 94.2±4.5 | 93.9±4.4 | 90.0±5.5 ◁ | 92.7±4.9 | 90.7±5.8 ◁ | 89.7±6.1 ◁ |
| Wine | 98.0±1.6 | 98.0±1.6 | 97.3±2.0 ◁ | 97.5±1.7 | 97.3±2.0 | 96.6±2.2 ◁ |
| Glass | 65.2±7.7 | 60.6±8.6 ◁ | 59.5±8.0 ◁ | 63.5±8.0 | 58.9±8.0 ◁ | 57.9±7.5 ◁ |
| SVMguide2 | 76.3±4.1 | 74.6±4.2 ◁ | 72.8±4.1 ◁ | 75.6±4.3 | 73.8±4.4 ◁ | 71.9±4.5 ◁ |
| | $\eta = 10\%$ | | | $\eta = 20\%$ | | |
| New-thyroid | 92.3±5.4 | 89.0±5.5 ◁ | 88.3±6.6 ◁ | 89.5±6.0 | 85.9±7.4 ◁ | 85.7±7.7 ◁ |
| Wine | 97.0±2.2 | 96.4±2.6 | 95.6±4.6 ◁ | 96.6±2.7 | 95.5±2.6 ◁ | 95.1±3.0 ◁ |
| Glass | 63.9±7.9 | 58.0±7.4 ◁ | 55.7±7.7 ◁ | 59.7±8.3 | 55.5±7.3 ◁ | 52.8±7.8 ◁ |
| SVMguide2 | 74.9±4.4 | 72.8±4.7 ◁ | 71.5±4.7 ◁ | 72.8±5.1 | 71.4±5.0 ◁ | 67.5±5.6 ◁ |

Table 2 displays for each problem the average BCR of each method for the different values of $\eta$ considered. When the performance of a method is significantly different from the performance of RMGPC, as estimated by a Wilcoxon rank test (p-value $< 1\%$), the corresponding BCR is marked with the symbol ◁. The table shows that, when there is no noise in the labels (*i.e.*, $\eta = 0\%$), RMGPC performs similarly to SMGPC in *New-Thyroid* and *Wine*, while it outperforms SMGPC in *Glass* and *SVMguide2*. As the level of noise increases, RMGPC is found to outperform SMGPC in all the problems investigated. HTPC typically performs worse than RMGPC and SMGPC independently of the value of $\eta$. This can be a consequence of HTPC using the Laplace approximation for approximate inference [9]. In particular, there is evidence indicating that the Laplace approximation performs worse than EP in the context of Gaussian process classifiers [15]. Extra experiments comparing RMGPC, SMGPC and HTPC under 3 different noise scenarios appear in the supplementary material. They further support the better performance of RMGPC in the presence of outliers in the data.

## 4.2 Outlier Identification

A second batch of experiments shows the utility of RMGPC to identify data instances that are likely to be outliers. These experiments use the *Glass* dataset from the previous section. Recall that for this

dataset RMGPC performs significantly better than SMGPC for $\eta = 0\%$, which suggest the presence of outliers. After normalizing the *Glass* dataset, we run RMGPC on the whole data and estimate the posterior probability that each instance is an outlier using (21). The hyper-parameters of RMGPC are estimated as described in the previous section. Figure 1 shows for each instance $(\mathbf{x}_i, y_i)$ of the *Glass* dataset, with $i = 1, \ldots, n$, the value of $\mathcal{P}(z_i = 1|\mathbf{y}, \mathbf{X})$. Note that most of the instances are considered to be outliers with very low posterior probability. Nevertheless, there is a small set of instances that have very high posterior probabilities. These instances are unlikely to stem from (1) and are expected to be misclassified when placed on the test set. Consider the set of instances that are more likely to be outliers than normal instances (*i.e.*, instances $3, 36, 127, 137, 152, 158$ and $188$). Assume the experimental protocol of the previous section. Table 3 displays the fraction of times that each of these instances is misclassified by RMGPC, SMGPC and HTPC when placed on the test set. The posterior probability that each instance is an outlier, as estimated by RMGPC, is also reported. The table shows that all the instances are typically misclassified by all the classifiers investigated, which confirms the difficulty of obtaining accurate predictions for them in practice.

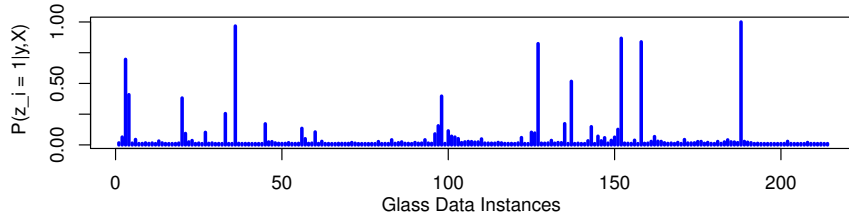

Figure 1: Posterior probability that each data instance form the *Glass* dataset is an outlier.

Table 3: Average test error in % of each method on each data instance that is more likely to be an outlier. The probability that the instance is an outlier, as estimated by RMGPC, is also displayed.

| | | **Glass Data Instances** | | | | | | |
|---|---|---|---|---|---|---|---|---|
| | | 3-rd | 36-th | 127-th | 137-th | 152-th | 158-th | 188-th |
| Test Error | RMGPC | 100.0±0.0 | 100.0±0.0 | 100.0±0.0 | 100.0±0.0 | 100.0±0.0 | 100.0±0.0 | 100.0±0.0 |
| | SMGPC | 100.0±0.0 | 92.0±5.5 | 100.0±0.0 | 100.0±0.0 | 100.0±0.0 | 100.0±0.0 | 100.0±0.0 |
| | HTPC | 100.0±0.0 | 84.0±7.5 | 100.0±0.0 | 100.0±0.0 | 100.0±0.0 | 100.0±0.0 | 100.0±0.0 |
| $\mathcal{P}(z_i = 1|\mathbf{y}, \mathbf{X})$ | | 0.69 | 0.96 | 0.82 | 0.51 | 0.86 | 0.83 | 1.00 |

# 5 Conclusions

We have introduced a Robust Multi-class Gaussian Process Classifier (RMGPC). RMGPC considers only the number of errors made, and not the distance of such errors to the decision boundaries of the classifier. This is achieved by introducing binary latent variables that indicate when a given instance is considered to be an outlier (wrongly labeled instance) or not. RMGPC can also identify the training instances that are more likely to be outliers. Exact Bayesian inference in RMGPC is intractable for typical learning problems. Nevertheless, approximate inference can be efficiently carried out using expectation propagation (EP). When EP is used, the training cost of RMGPC is $\mathcal{O}(ln^3)$, where $l$ is the number of classes and $n$ is the number of training instances. Experiments in four multi-class classification problems show the benefits of RMGPC when labeling noise is injected in the data. In this case, RMGPC performs better than other alternatives based on considering latent Gaussian noise or noise which follows a distribution with heavy tails. When there is no noise in the data, RMGPC performs better or equivalent to these alternatives. Our experiments also confirm the utility of RMGPC to identify data instances that are difficult to classify accurately in practice. These instances are typically misclassified by different predictors when included in the test set.

**Acknowledgment**

All experiments were run on the Center for Intensive Computation and Mass Storage (Louvain). All authors acknowledge support from the Spanish MCyT (Project TIN2010-21575-C02-02).

## Footnotes

[1]The supplementary material is available online at http://arantxa.ii.uam.es/%7edhernan/RMGPC/.

[2]The R source code of RMGPC is available at http://arantxa.ii.uam.es/%7edhernan/RMGPC/.

# References

[1] Carl Edward Rasmussen and Christopher K. I. Williams. *Gaussian Processes for Machine Learning (Adaptive Computation and Machine Learning)*. The MIT Press, 2006.

[2] Christopher K. I. Williams and David Barber. Bayesian classification with Gaussian processes. *IEEE Transactions on Pattern Analysis and Machine Intelligence*, 20(12):1342–1351, 1998.

[3] Hyun-Chul Kim and Zoubin Ghahramani. Bayesian Gaussian process classification with the EM-EP algorithm. *IEEE Transactions on Pattern Analysis and Machine Intelligence*, 28(12):1948–1959, 2006.

[4] R.M Neal. Regression and classification using Gaussian process priors. *Bayesian Statistics*, 6:475–501, 1999.

[5] Matthias Seeger and Michael I. Jordan. Sparse Gaussian process classification with multiple classes. Technical report, University of California, Berkeley, 2004.

[6] M. Opper and O. Winther. Gaussian process classification and SVM: Mean field results. In P. Bartlett, B.Schoelkopf, D. Schuurmans, and A. Smola, editors, *Advances in large margin classifiers*, pages 43–65. MIT Press, 2000.

[7] Daniel Hernández-Lobato and José Miguel Hernández-Lobato. Bayes machines for binary classification. *Pattern Recognition Letters*, 29(10):1466–1473, 2008.

[8] Hyun-Chul Kim and Zoubin Ghahramani. Outlier robust Gaussian process classification. In *Structural, Syntactic, and Statistical Pattern Recognition*, volume 5342 of *Lecture Notes in Computer Science*, pages 896–905. Springer Berlin / Heidelberg, 2008.

[9] Fabian L. Wauthier and Michael I. Jordan. Heavy-Tailed Process Priors for Selective Shrinkage. In J. Lafferty, C. K. I. Williams, R. Zemel, J. Shawe-Taylor, and A. Culotta, editors, *Advances in Neural Information Processing Systems 23*, pages 2406–2414. 2010.

[10] Thomas Minka. *A Family of Algorithms for approximate Bayesian Inference*. PhD thesis, Massachusetts Institute of Technology, 2001.

[11] A. Asuncion and D.J. Newman. UCI machine learning repository, 2007.

[12] Chih-Chung Chang and Chih-Jen Lin. *LIBSVM: A library for support vector machines*, 2001.

[13] Christopher M. Bishop. *Pattern Recognition and Machine Learning (Information Science and Statistics)*. Springer, August 2006.

[14] T. Minka and J. Lafferty. Expectation-propagation for the generative aspect model. In Adnan Darwiche and Nir Friedman, editors, *Proceedings of the 18th Conference on Uncertainty in Artificial Intelligence*, pages 352–359. Morgan Kaufmann, 2002.

[15] Malte Kuss and Carl Edward Rasmussen. Assessing approximate inference for binary Gaussian process classification. *Journal of Machine Learning Research*, 6:1679–1704, 2005.

[16] H Nickisch and CE Rasmussen. Approximations for binary Gaussian process classification. *Journal of Machine Learning Research*, 9:2035–2078, 10 2008.

[17] Marcel Van Gerven, Botond Cseke, Robert Oostenveld, and Tom Heskes. Bayesian source localization with the multivariate Laplace prior. In Y. Bengio, D. Schuurmans, J. Lafferty, C. K. I. Williams, and A. Culotta, editors, *Advances in Neural Information Processing Systems 22*, pages 1901–1909, 2009.

[18] Matthias Seeger. Expectation propagation for exponential families. Technical report, Department of EECS, University of California, Berkeley, 2006.

